# Accurate and Steady Inertial Pose Estimation through Sequence Structure Learning and Modulation

Yinghao Wu[1]     Chaoran Wang[1]     Lu Yin[1]     Shihui Guo[1*†]     Yipeng Qin[2]

[1]School of Informatics, Xiamen University, China
[2]School of Computer Science & Informatics, Cardiff University, UK

## Abstract

Transformer models excel at capturing long-range dependencies in sequential data, but lack explicit mechanisms to leverage structural patterns inherent in fixed-length input sequences. In this paper, we propose a novel sequence structure learning and modulation approach that endows Transformers with the ability to model and utilize such fixed-sequence structural properties for improved performance on inertial pose estimation tasks. Specifically, our method introduces a Sequence Structure Module (SSM) that utilizes structural information of fixed-length inertial sensor readings to adjust the input features of transformers. Such structural information can either be acquired by learning or specified based on users' prior knowledge. To justify the prospect of our approach, we show that i) injecting spatial structural information of IMUs/joints learned from data improves accuracy, while ii) injecting temporal structural information based on smooth priors reduces jitter (i.e., improves steadiness), in a spatial-temporal transformer solution for inertial pose estimation. Extensive experiments across multiple benchmark datasets demonstrate the superiority of our approach against state-of-the-art methods and has the potential to advance the design of the transformer architecture for fixed-length sequences.

## 1   Introduction

Estimating human pose is a long-standing and prominent task that underlies many computer vision and graphics applications, e.g., animation production, virtual reality. As an alternative to vision-based solutions, wearable device-based methods are gaining increasing interest as they are environment-free, occlusion-unaware, privacy-friendly. For ensuring high accuracy and maintaining portability and usability, most prior works [18, 55, 56, 20, 60] abandon densely placed configurations, instead leveraging a sparse set of Inertial Measurement Units (IMUs) to reconstruct human motion. Since IMUs can provide continuous measurements of rotation and acceleration, we define pose estimation with sparse inertial sensors as a *sequence learning* task.

Recently, Transformer-based architectures have achieved tremendous success in various sequence learning tasks, and applying them to sparse inertial motion capture is a natural idea. However, empirically, we find that, directly using the native transformer to model IMU sequences results in unacceptable jitter and inaccurate postures. Through our analysis, we attribute this to *the native Transformer architecture, whose self-attention mechanism was originally designed to flexibly handle variable-length sequence inputs, thus lacks inductive bias for modeling fixed-length sequences that have clear structures*. For instance, for an IMUs reading sequence, the length is usually fixed (e.g., the number of observed past frames in a time window or the number of IMUs) and each token in the sequence has a specific meaning (e.g., each temporal token denotes a frame and each spatial token

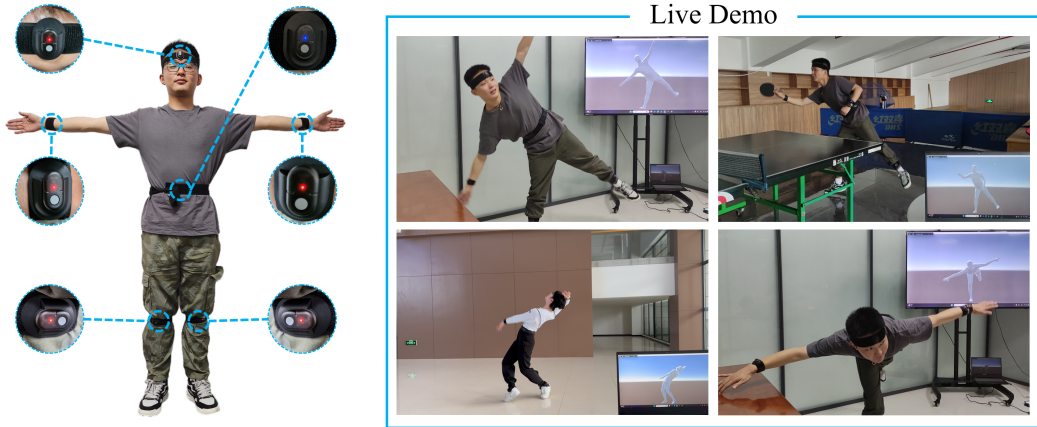

Figure 1: Left: we use only six IMUs to predict the full-body pose in real-time, which are fixed on left and right forearm, the left and right lower leg, the head, and the pelvis. Right: our system is capable of capturing a wide range of daily motions as well as challenging movements.

represents an IMU). In other words, as a representative type of fixed-length sequences, IMU readings have clear structures both in spatial and temporal dimensions. However, such structural information is not explicitly modeled in native Transformers.

To bridge this gap, we present an innovative approach for learning and modulating sequence structure, which empowers transformers to effectively capture and leverage structural properties of fixed-length sequences, leading to enhanced performance in inertial pose estimation tasks. Specifically, our method introduces a novel Sequence Structure Module (SSM) designed to leverage the structural information of fixed-length sequences to adjust the input features of Transformers. For inertial pose estimation tasks, we devised two SSM variants: SSM-S and SSM-T, for injecting spatial and temporal structural information into spatial and temporal transformer features, respectively. Extensive experiments on four public benchmarks demonstrate that our method achieves superior performances than state-of-the-art methods, where the average errors of the whole-body angles decreased by 13% and 38%, together with the lowest jitter, on the DIP-IMU [18] and TotalCapture [46] datasets, respectively. In addition, we implemented a real-time pose estimation system to test the performance of our approach in real-world scenarios. In summary, our contributions include:

- We identify a key limitation of the native transformer architecture: its lack of inductive biases for modeling fixed-length sequences with inherent structural properties. To address this shortcoming, we propose a novel Sequence Structure Module (SSM) that enables transformers to effectively capture and leverage the structural priors present in fixed-length sequential data.

- For inertial motion capture tasks involving sequential IMU data, we propose two SSM variants: SSM-S and SSM-T, which incorporate structural inductive biases of the IMU sensor layout (spatial) and time frames (temporal), respectively, into transformer learning.

- Extensive experiments demonstrate that our method outperforms state-of-the-art ones on the DIP-IMU and TotalCapture datasets by a large margin. To further demonstrate the superiority of our approach, we implemented a *real-time motion capture system* based on six IMUs to evaluate the performance of our model in complex *real-world scenarios*.

## 2 Related Works

### 2.1 Human Pose Estimation

Human pose estimation (HPE) has been a long-standing research topic, with numerous researchers exploring it using various types of sensors. As our research lies on IMU sensors, we roughly categorize sensors into two types: non-IMU and IMU.

**HPE with non-IMU Sensors**. In non-IMU human motion capture solutions, vision-based methods still dominate the mainstream so far. Early methods [14, 23, 16, 42] used multiple cameras and marker points to capture human poses, which imposed significant constraints on the environment. With the popularity of deep learning, an increasing number of methods are using a single camera to capture human 2D/3D poses, such as CPN [8], HRNet [44] and others [62, 22, 54, 51, 61, 45, 39, 28, 67, 58, 25], achieving significant success. In addition, there are also other types of sensors used for tracking human motion, such as 6DOF trackers [43, 5, 13, 19, 63], flexible fabric sensors [7, 26, 68], wireless sensors [6, 53] and hybrid sensors [3, 35, 38]. Although each method has its own advantages, motion capture systems based on IMUs are emerging and gaining prominence, due to their wearability, portability, and ease of use.

**HPE with IMU Sensors**. The advantage of inertial motion capture systems compared to vision-based methods is their resistance to extreme conditions such as bright lights and occlusion. Commercial systems like Xsens [41] and Noitom [36] place multiple IMUs on the user's body, while achieving accurate pose estimation, also restricting the user's movements and requiring lots of time for IMU attachment. To enhance user comfort and usability, an increasing number of studies [48, 18, 56, 55, 20, 60] are shifting towards sparse inertial motion capture and predict human postures using only a few IMUs. As a pioneering work in sparse inertial pose prediction, SIP [48] demonstrates that recovering full-body motion using only 6 IMUs is feasible for the first time, but it is an optimization-based approach which is computationally slow. Huang et al. [18], the first to introduce neural networks into this task, employ a bidirectional RNN to achieve real-time performance. After that, Transpose [56] proposes a multi-stage pose estimation framework, utilizing three sub-networks based on bidirectional RNN to predict pose, further enhancing accuracy. However, both of [18, 56] require future frames as input, which adds additional latency. PIP [55], introduces physical constraints, improving the physical plausibility of motion without the need for future frames. Another work, TIP [20], utilizes Transformer to predict human body poses while simultaneously generating terrain, achieving prediction of human motion in non-planar environments. Contemporaneous work, DynaIP [60], leverages pseudo-velocity learning to fully utilize acceleration and models the human body into three separate regions, each focusing on their unique characteristics. In addition, some studies [34, 69] have attempted to place IMUs in objects carried by the body for tracking human motion. For example, Zuo et al. [69] use a loose-wear jacket with 4 IMUs to capture the upper body motion, which provide the users with high comfort and freedom of movement.

However, existing methods only focus on modeling the temporal dimension (whether using RNNs or Transformers) while neglecting the spatial dimension. Unlike them, we utilize a two-stage Transformer-based spatial-temporal framework, independently capturing the dependencies of both space and time. Meanwhile, we have also designed two modules, SSM-S and SSM-T, enabling the Transformer to more effectively leverage structural information from fixed-length sequences.

## 2.2 Transformer Variants for Time Series

Transformer has seen a number of modifications to address the limitations of well-known works such as BERT [11] and ViT [12]. In the field of time-series data modeling, researchers have proposed various approaches, one common method being modifications in positional encoding [24, 59, 27, 64, 50, 65]. For example, Transformer-XL [10] introduced relative positional encoding, enabling the model to capture long-range dependencies, TCN-Transformer [2] combines the characteristics of convolutional networks with relative positional encoding. Self-attention module is the central part of Transformer. However, for many long-sequence based tasks, the time complexity of self-attention module is a computational bottleneck. Various works [31, 64, 65, 49, 4, 52] are proposed to address this issue. Longformer [4] employs a sparse attention mechanism, specifying local and global attention, allowing the model to handle long sequences while maintaining computational efficiency. Linformer [49] approximates the original high-dimensional attention matrix through low-rank projection, reducing both computational and memory requirements. Additionally, some researchers made structural modifications [52, 10, 64, 21] to Transformer for time series tasks. For example, Transformer-XL [10] incorporates a segment-level recurrence mechanism in the encoder to handle longer contextual information. Reformer [21] utilizes locality-sensitive hashing (LSH) and a reversible network structure, enabling the model to process extremely long sequences. Informer [64] introduces ProbSparse Self-Attention, reducing the computational complexity of self-attention.

Previous studies have modified the Transformer extensively, but they have largely overlooked its limitations in modeling fixed-length sequences and does not impose Transformer to explicitly utilize the inherent structural information within it. Our work bridges this gap.

## 3 Method

### 3.1 Why is Sequence Structure Modeling Missing in Native Transformer Architecture?

The Transformer architecture [47] was originally designed to accomplish machine translation tasks in natural language processing. To handle variable-length textual inputs with different syntaxes, the native Transformer architecture does not make any inductive bias on their structures, but instead focuses on the content of the input text. Specifically, let $X \in \mathbb{R}^{N \times d}$ be the embedded input sequence of length $N$ and feature dimension $d$, the self-attention mechanism is defined as:

$$\text{Attention}(Q, K, V) = \alpha V = \text{softmax}(QK^\top/\sqrt{d})V \tag{1}$$

where $Q = XW_Q$, $K = XW_K$ and $V = XW_V \in \mathbb{R}^{N \times d}$; $W_Q, W_K$, and $W_V \in \mathbb{R}^{d \times d}$; $\alpha \in \mathbb{R}^{N \times N}$. Among them, only the attention matrix $\alpha$ is modeling the relationships among the $N$ input tokens. However, its element $\alpha_{(i,j)}$ is calculated as the product between the $i$-th query ($Q$) and the $j$-th key ($K$) pair, thus representing the relationship between individual tokens rather than the structure of the input sequence as a whole. This enables it to handle sequences with different $N$ that do not share a common structure (e.g., sentences).

However, in many other domains (e.g., pose estimation), the input sequences usually have fixed length (e.g., the number of observed past frames or body joints) and a clear structure (e.g., temporal continuity or spatial relationship), which implies that the structural information of input sequences can facilitate learning. Motivated by this key insight, we propose a novel Sequence Structure Module (SSM) to endow the Transformer architecture with the ability to model and utilize the structural information inherent in fixed-length input sequences.

### 3.2 Sequence Structure Module

Our Sequence Structure Module (**SSM**) aims to fully utilize the structured information of fixed-length sequence inputs to compensate for the lack of inductive bias in the transformer architecture. Specifically, as shown in Fig. 2 (d), given a sequence embedding $X \in \mathbb{R}^{N \times d}$, before entering the transformer encoder, we multiply $X$ with a **structural matrix** $S \in \mathbb{R}^{N \times N}$, followed by a LayerNorm (LN) layer [1] and a MLP Block:

$$\widetilde{X} = \text{MLP}(\text{LN}(SX)) \tag{2}$$

where $\widetilde{X} \in \mathbb{R}^{N \times d}$, $\text{LN}(\cdot)$ is used to regularize the model and maintain gradient stability during training and $\text{MLP}(\cdot)$ is used to increase the capacity of the module. *The key distinction between the structural matrix $S$ and the self-attention matrix $\alpha$ is that the elements of $S$ are independent of the input $X$, making $S$ universally applicable to all input sequences regardless of their content. This allows $S$ to effectively capture structural information.* Then, we feed the structure-enhanced $\widetilde{X}$ into the subsequent transformer encoder for modeling the long-range dependencies using Eq. (1). The structural matrix $S$ can be obtained in various ways. For example, it can come from prior knowledge in the specific domains, be entirely data-driven through learning, or be a combination of the two. Here, we categorize the structure into the following three types: **Explicit Structure** $S_E$, **Implicit Structure** $S_I$, and **Explicit-Implicit Hybrid Structure** $S_{EI}$.

**Explicit Structure (ES)** is particularly useful when each token in the input sequence of a fixed length $N$ has a clear meaning (e.g., each token in a time sequence represents a frame), and the structural relationships between the $N$ tokens can be precisely captured based on prior knowledge. That is, each element $S_E(i, j)$ in the structural matrix is provided by the user before training, allowing the user to impose prior knowledge of the sequence structure on learning. Notably, when $S_E$ is the identity matrix $I$, no modifications are made and the sequence structure module degenerates. We provide two examples of how to construct explicit structures $S_E$ in Sec. 3.4.

**Implicit Structure (IS)** becomes appropriate when the meaning of each token in the sequence and/or the structural relationships between tokens are unclear. Unlike **ES**, which are completely determined

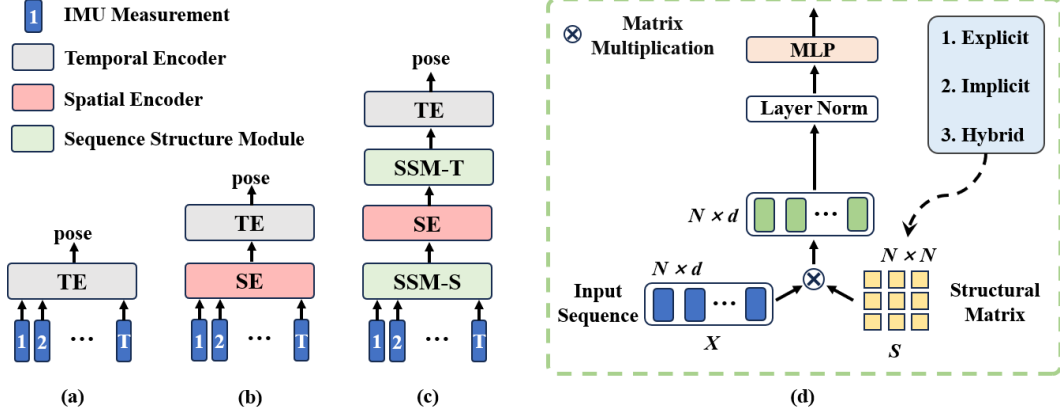

Figure 2: (a) Previous work only employ temporal encoders (RNN or transformer) to predict pose. (b) Our spatial-temporal framework. (c) Our spatial-temporal framework with **SSM**. (d) Our sequence structure module (**SSM**), simply consists of structural matrix $S$, LayerNorm [1] and MLP Block.

by the user, the establishment of **IS** relies on a learnable matrix $P$, that is learned in a data-driven manner. We define the implicit structure matrix $S_I$ using the following equation:

$$S_I = I + P \tag{3}$$

where $I, P \in \mathbb{R}^{N \times N}$ and $I$ denotes the identity matrix. Notably, another way to define $S_I$ is to solely use $P$ without the identity matrix $I$, namely $S_I = P$. But as pointed out in [17], in the extreme situation, it is easier to optimize $P$ into a zero matrix rather than an identity matrix.

**Explicit-Implicit Hybrid Structure (EIHS)** aims to strike a balance between **ES** and **IS** by allowing the user to provide an initial structure matrix based on prior knowledge, while also using data-driven methods to modify it. It can also be referred to as a non-identity matrix initialization of Eq. (3):

$$S_{EI} = S_E + P \tag{4}$$

where $S_E \in \mathbb{R}^{N \times N}$ represents the explicit structure, and $P \in \mathbb{R}^{N \times N}$ is a learnable matrix. In the extreme case, when the user-designed explicit structure $S_E = I$, **EIHS** degenerates into **IS**.

### 3.3 SSM for Transformer-based Sparse Inertial Pose Estimation

**Problem Formulation**. Our task is to accomplish real-time human pose estimation using data acquired from 6 inertial sensors positioned on the wrists of both hands, ankles of both feet, waist, and head (Fig. 1). Each IMU can provide sequential acceleration $\mathcal{A}$ and orientation $\mathcal{R}$ signals on the body part it is placed on, where $\mathcal{A} \in \mathbb{R}^3$ is the linear acceleration and $\mathcal{R} \in \mathbb{R}^{3 \times 3}$ is the rotation matrix. Our goal is to learn a mapping $f$ which reconstructs the joints' rotations of the full body:

$$\mathcal{O}_{1:J}^T = f(\{\mathcal{A}, \mathcal{R}\}_{1:N}^{1:T}) \tag{5}$$

where $T$ denotes the number of observed frames from the past, $J$ denotes the number of predicted joints, $N$ denotes the number of IMUs, and $\mathcal{O} \in SO(3)$ is the rotation of body joints, representing the human pose with a certain skeleton (e.g. SMPL [29]).

**Spatial-Temporal Transformer with SSM**. To accomplish this task, unlike previous works that only employed temporal encoders (Fig. 2 (a)), we utilize a spatial-temporal framework transformer network as our baseline model (Fig. 2 (b)), where the spatial transformer models the local motion correlations among $N$ IMUs/joints within a frame, while the temporal transformer captures the global dependencies between $T$ frames throughout the entire sequence. Since the values of $N$ and $T$ are typically fixed, (e.g. $N = 6$ and $T = 30$), we introduce two variants of Sequence Structure Module, **SSM-S** and **SSM-T**, to leverage the structural information of fixed sequences in spatial and temporal dimensions, respectively (Fig. 2 (c)). As previously mentioned, each SSM has three different choices for the structure matrix $S$. After thorough experimental comparisons in Sec 4.4, our final choice is that, the structural matrix of SSM-S is derived from **EIHS**, and the structure of SSM-T is **ES**. Due to page limitations, we have included more network details in Appendix Sec. 7.

### 3.4 Constructing Explicit Structures

The power of SSM lies in its large design space of explicit structures $S_E$. Here, we take the sparse inertial pose estimation task as an example, to demonstrate how to construct $S_E$ in spatial and temporal dimensions based on prior knowledge.

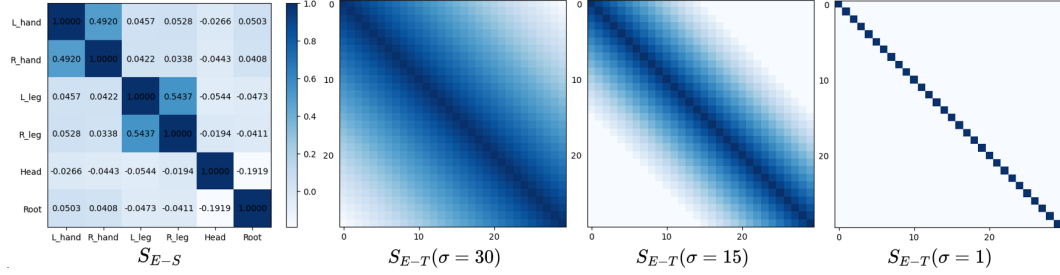

Figure 3: The visualization of $S_{E-S}$, and $S_{E-T}$ with different $\sigma$.

**Spatial Structure Construction**. The motivation for constructing the spatial structure $S_{E-S}$ is to explore relevance and consistency between body joints when the human body moves. "Relevance and consistency" refer to the tendency for certain joints to exhibit similar movement patterns across a large sample of human motions. For instance, when the left hand moves forward, the head, torso, and legs are likely to move in a certain coordinated manner.

We utilize a statistical approach to accomplish this task. Specifically, we utilize the AMASS [32] dataset to decompose the rotations of different joints in each frame into rotations along the $x$-axis, $y$-axis, and $z$-axis, representing the rotations in terms of Euler angles. By doing so, we obtain $R^x, R^y, R^z \in \mathbb{R}^{J \times f}$, where $f$ denotes the number of observed frames and $J$ denotes the number of joints. We select subsets ($sub$) of the rotations consisting of the $N$ joints where IMUs are placed (e.g., $N = 6$) and have: $R^{(x,sub)}, R^{(y,sub)}, R^{(z,sub)} \in \mathbb{R}^{N \times f}$. We independently calculate the correlation matrix $C^k \in \mathbb{R}^{N \times N}, k = x, y, z$ for each subset and sum up the results for average to obtain the spatial explicit structure $S_{E-S} \in \mathbb{R}^{N \times N}$:

$$C^k(i,j) = \frac{\text{cov}(R^{(k,sub)}_{(i)}, R^{(k,sub)}_{(j)})}{\sqrt{\text{var}(R^{(k,sub)}_{(i)}) \times \text{var}(R^{(k,sub)}_{(j)})}} \tag{6}$$

$$S_{E-S} = \frac{1}{3} \sum_{k=x,y,z} C^k \tag{7}$$

where cov denotes the covariance between two variables, and var denotes the variance of a variable. The resulting $S_{E-S}$ is shown in Fig. 3. It can be observed that, the movements between the two hands and the two legs exhibit high correlation; but the movements of the head and the root (spine) are relatively independent; which aligns with our intuition.

**Temporal Structure Construction**. Unlike the use of statistical methods to calculate spatial explicit structures $S_{E-S}$, the construction of temporal explicit structures $S_{E-T}$ is much more straightforward, which follows the "distance" between two frames within a time sequence. That is, when two frames are "close" enough, such as adjacent frames, their correlation is relatively high; when the frames are "farther apart", such as the first and last frames in a sequence, their correlation tends to decrease as the "distance" increases; akin to a smoothness prior. Mathematically, we define $S_{E-T} \in \mathbb{R}^{T \times T}$ using a function that linearly decreases with increasing "distance":

$$S_{E-T}(i,j) = \begin{cases} 0 & \text{if } |i-j| \geq \sigma, \\ 1 - \frac{|i-j|}{\sigma} & \text{else} \end{cases} \tag{8}$$

where $\sigma$ is a hyperparameter, which represents the maximum "distance" between two frames that are considered correlated. Specifically, when the "distance" between $i$-th and $j$-th frame, namely $|i-j| \geq \sigma$, we consider these two frames to be unrelated. Fig. 3 shows the visualization of $S_{E-T}$

under different values of $\sigma$. In our experimental setup, $T = 30$ and $\sigma = 10$. For details on how to determine $\sigma$, please refer to Sec. 4.4.

Notably, in addition to the two methods mentioned above, there are many ways to construct explicit structures. In fact, any matrix with a specific meaning can serve as an explicit structure. While we have presented two illustrative examples, the ongoing quest to design tailored SSM explicit structures for diverse tasks offers boundless opportunities for exploration and innovation.

## 4 Experiments

### 4.1 Datasets and Evaluation Metrics

**Datasets.** We use the following datasets in our experiments, which can be divided into three categories: 1) Synthetic dataset: AMASS [32]. 2) Real datasets with SMPL [29] skeleton: DIP-IMU [18] and TotalCapture [46]. 3) Real datasets with Xsens [41] skeleton: AnDy [33], CIP [37], and Emokine [9]. We use them to train and evaluate our methods as follows: 1) Following [56, 55, 20], we first pre-train our model on AMASS and fine-tune it on the training set of DIP-IMU, then test it on the test set of DIP-IMU and the entire TotalCapture dataset; 2) Following [60], we train our model on the training sets of AnDy, CIP, and Emokine datasets and test it on the test sets of AnDy and CIP datasets.

**Evaluation Metrics.** We use the following metrics to evaluate our method: 1) *SIP error*, measuring the mean global rotation error of upper arms and upper legs in degrees; 2) *Angular error*, measuring the mean global rotation error of all body joints in degrees; 3) *Positional error*, measuring the mean Euclidean distance error of all estimated joints in centimeters with the root joint (Spine) aligned; 4) *Mesh error*, measuring the mean Euclidean distance error of all vertices of the estimated body mesh with the root joint (Spine) aligned. The vertex coordinates are calculated by applying the pose parameters to the SMPL [29] body model; 5) *Jitter*, measuring the mean jerk (time derivative of acceleration) of all body joints in the global space, which reflects the smoothness of the motion [15].

### 4.2 Implementation Details

We implement our method using the PyTorch [40] framework on one NVIDIA GeForce RTX 4090 GPU. PyTorch version is 2.0.0, and CUDA version is 11.8. During the training stage, we use the AdamW [30] optimizer to train our model with a batch size of 4096. The learning rate is initialized to 0.0001 and decayed by 0.99 per epoch. We implement the live demo using a laptop equipped with an Intel® Core™ i9-13900HX Processor CPU and an NVIDIA GeForce RTX 4060 GPU.

### 4.3 Comparisons with SOTA

**Quantitative Results.** We compare our method with state-of-the-art ones, including TransPose [56], TIP [20], PIP [55], DynaIP [60], which also accomplish pose estimation from only 6 IMUs signals. All metrics are calculated in the real-time setting and the best and runner-up results in each column are marked in **bold** and underline respectively.

Table 1: Comparison with SOTA methods on DIP-IMU [18] and TotalCapture [46] datasets with SMPL [29] skeleton. **Bold** indicates best and underline indicates runner-up results.

|  | DIP-IMU | | | | | TotalCapture | | | | |
|---|---|---|---|---|---|---|---|---|---|---|
|  | SIP Err | Ang Err | Pos Err | Mesh Err | Jitter | SIP Err | Ang Err | Pos Err | Mesh Err | Jitter |
| DIP[18] | 17.10 | 15.16 | 7.33 | 8.96 | 3.01 | 18.62 | 17.22 | 9.42 | 11.22 | 3.62 |
| Transpose[56] | 17.03 | 8.86 | 6.03 | 7.14 | 1.08 | 16.40 | 12.77 | 6.42 | 7.20 | 1.83 |
| TIP[20] | 16.92 | 9.07 | 5.63 | 6.62 | 1.53 | 13.20 | 12.24 | 5.68 | 6.78 | 1.57 |
| PIP[55] | 15.02 | 8.72 | 5.01 | 6.02 | 0.14 | 12.93 | 12.04 | 5.61 | 6.51 | 0.18 |
| DynaIP[60] | 14.11 | 7.00 | 4.97 | 5.97 | 0.18 | 12.42 | 11.06 | 5.11 | 5.79 | 0.22 |
| PNP[57] | 13.71 | 8.75 | 4.97 | 5.77 | 0.17 | 10.89 | 10.45 | 4.74 | 5.45 | 0.26 |
| Ours | **7.90** | **6.06** | **3.12** | **3.78** | **0.07** | **7.00** | **6.82** | **3.36** | **4.00** | **0.09** |

As shown in Table 1 and Table 2, the results indicate that our method has surpassed previous approaches by a significant margin on both four benchmark datasets, achieving more accurate and steadier pose estimation. Specifically, our SIP Err on four datasets is significantly ahead of other

Table 2: Comparison with SOTA methods on AnDy [33] and CIP [37] datasets with Xsens [41] skeleton.

| | AnDy | | | CIP | | |
|---|---|---|---|---|---|---|
| | SIP Err | Ang Err | Pos Err | SIP Err | Ang Err | Pos Err |
| Transpose[56] | 12.15 | 6.29 | 4.91 | 20.06 | 8.75 | 6.86 |
| TIP[20] | 10.11 | 4.55 | 3.56 | 13.05 | 5.67 | 4.30 |
| PIP[55] | 9.49 | 4.09 | 3.29 | 12.68 | 5.52 | 4.12 |
| DynaIP[60] | 8.93 | 3.45 | 3.41 | 11.42 | 4.54 | 3.69 |
| Ours | 4.56 | 3.37 | 1.73 | 8.14 | 5.49 | 2.57 |

Table 3: Ablation study of SSM-S and SSM-T.

| Models | Ang Err | Jitter | $\tau$ |
|---|---|---|---|
| Baseline | 8.82 | 0.48 | 14.25 |
| + SSM-S | 7.83 | 0.43 | 12.04 |
| + SSM-T | 7.93 | 0.09 | 8.68 |
| Ours | 6.82 | 0.09 | 7.46 |

methods, outperforming the runner-up by 44%, 44%, 49% and 29% respectively. We attribute this to our spatio-temporal framework and the spatial structure information in SSM-S, which help the model better capture the motion correlation between body joints. Besides, on the DIP-IMU and TotalCapture datasets, our predicted motion sequences exhibit the least jitter (i.e., smoothest motion sequence). We attribute this to the temporal sequence structure information in our SSM-T, which provides more temporal prior knowledge between frames and smooths the input features to generate more stable and consistent motion prediction results.

**Qualitative Results.** We also provide a visual comparison between the estimated pose and the ground truth on the TotalCapture dataset. Compared with state-of-the-art methods, our method achieves more precise predictions as shown in Fig. 4. The comparison of the two actions (leaning forward and bending over) indicates that, our method can estimate the positions of arms and legs more accurately than previous methods. Additionally, the comparison of the two ambiguous actions (raising a leg and raising both hands) in the second row shows that our model better identifies these ambiguous actions.

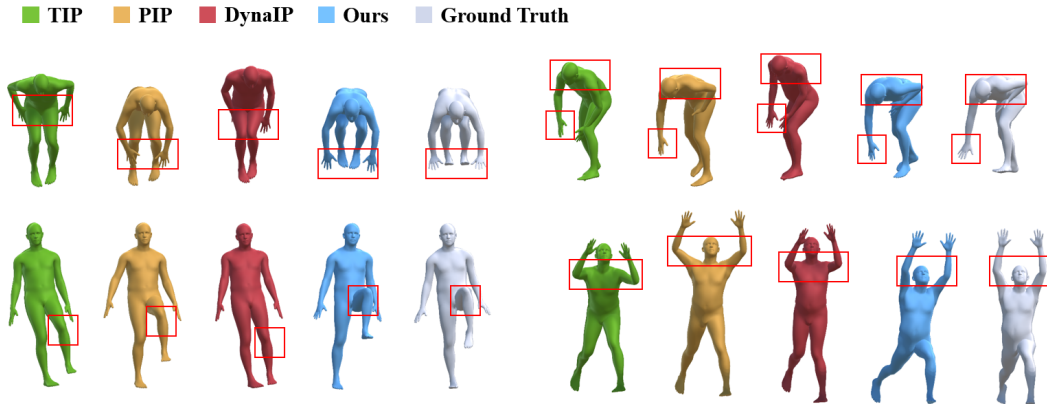

Figure 4: Qualitative comparisons with the state-of-the-art methods on TotalCapture dataset.

**Analysis for Error of Joints**. As shown in Fig. 5, we also compare the angular errors of individual joints on the DIP-IMU and TotalCapture datasets. It can be observed that, the errors of previous methods are mostly concentrated in the joints of the hands (L/R elbows and shoulders) and legs (L/R hips and knees), which are the main sources of *SIP Err*. We believe this is because they only capture the inter-frames dependencies in the temporal dimension, while neglecting to model the motion correlations between body joints in the spatial dimension. Unlike them, we utilize a two-stage spatial-temporal framework, where the spatial encoder independently models the motion consistency of IMUs/joints within a frame, and SSM-S injects spatial structural information into the features. Their combined effect allows for more accurate estimation of each joint rotation.

## 4.4 Ablation Study

We conduct ablation experiments on the TotalCapture dataset, reporting two metrics: *Ang Err* and *Jitter*. Additionally, we observe a trade-off between *Ang Err* and *Jitter*, challenging the simultaneous achievement of the lowest values for both. For convenience, we introduce a temporary metric $\tau = (Ang\,Err) * Exp(Jitter)$ as a reference for selecting the optimal model, where a lower $\tau$ represents the balance between accuracy and stability, with $Exp(\cdot)$ denoting exponential operation.

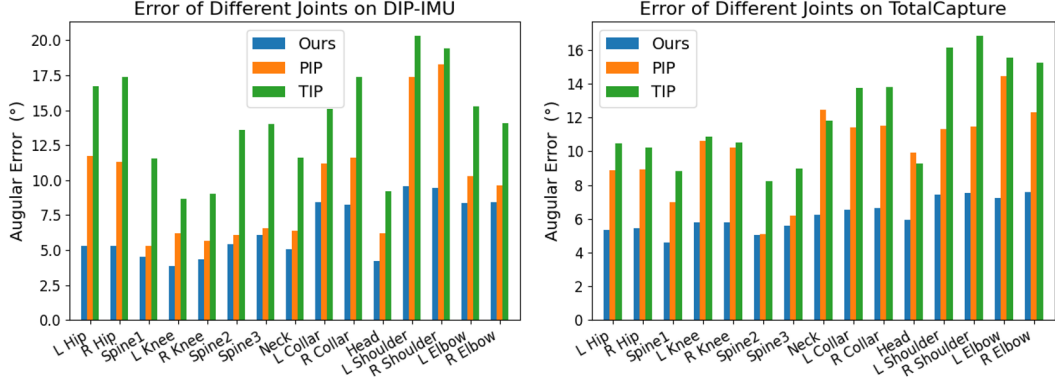

Figure 5: Error of different joints on DIP-IMU and TotalCapture datasets.

**Design of Sequence Structure for SSM-S and SSM-T**. We first explore the designs of sequence structure in SSM-S and SSM-T. As previously mentioned, each SSM can have three different choices, resulting in a total of $3 \times 3 = 9$ combinations for the two SSMs. To simplify this, we first fix the type of SSM-S as IS and explore the best structure type for SSM-T. As shown in Tab. 4, comparing Settings 1, 2, and 3 demonstrates that ES is the best structure choice for SSM-T, as the other two structures result in unacceptable jitter. Based on this, we conduct experiments with Settings 4 and 6. By comparing Settings 2, 4, and 6, we find that EIHS is the best choice for SSM-S, as it significantly reduces angular error without substantially increasing jitter (i.e., with lower $\tau$). Thus, we conclude that, EIHS + ES is the best combination for SSM-S and SSM-T.

Table 4: Ablation study on SSM design.

| Setting | SSM-S | SSM-T | Ang Err | Jitter | $\tau$ |
|---|---|---|---|---|---|
| 1 | IS | IS | 7.79 | 0.54 | 13.37 |
| 2 | IS | ES | 7.94 | 0.09 | 8.69 |
| 3 | IS | EIHS | 7.86 | 0.53 | 13.35 |
| 4 | ES | ES | 8.11 | 0.08 | 8.79 |
| 5 | EIHS | IS | 8.13 | 0.34 | 11.42 |
| 6 | EIHS | ES | **6.82** | **0.09** | **7.46** |

Table 5: Performance under different selection for hyperparameter $\sigma$.

| $\sigma$ | Ang Err | Jitter | $\tau$ |
|---|---|---|---|
| 30 | 8.08 | 0.07 | 8.66 |
| 25 | 8.05 | 0.07 | 8.63 |
| 20 | 7.76 | 0.07 | 8.32 |
| 15 | 7.85 | 0.08 | 8.50 |
| **10** | **6.82** | **0.09** | **7.46** |
| 5 | 7.33 | 0.13 | 8.34 |
| 1 | 7.25 | 0.38 | 10.60 |

**Further Exploration for** $S_{E-T}$. Fig. 3 shows the visualization of $S_{E-T}$ under different choices of $\sigma$, and Tab. 5 shows their performance on TotalCapture dataset. It can be observed that the model achieves the best performance when $\sigma = 10$. We hypothesize that this means the information from the adjacent 10 frames should be given more emphasis for IMU measurements in our case.

**Contribution of Each Component**. We conduct a thorough ablation study when $\sigma = 10$, to investigate the respective contributions of SSM-S and SSM-T. We use the most basic spatial-temporal framework (Fig. 2 (b)) as the baseline model and sequentially add SSM-S and SSM-T. The results are shown in Tab. 3. It can be observed that the primary function of SSM-S is to reduce joint angle error to improve the accuracy of human pose prediction, while the role of SSM-T is to reduce jitter to enhance the coherence of the posture and generate steadier motion sequence.

**In-depth analysis of SSM-S**. To look deeper into SSM-S, we visualized the spatial structure matrix $S_{E-S}$ (before training), the learnable matrix $P_S$ and the final spatial structure matrix $S_{EI-S}$ as shown in Fig. 6. It can be observed that:

- The overall pattern of the structure matrix remain the same before and after training ($S_{E-S}$ and $S_{EI-S}$), i.e., the movements between the two hands and the two legs still exhibit high correlation; and the movements of the head and the root (spine) are still negatively correlated. This demonstrates the effectiveness of our $S_{E-S}$ as initialization/prior.

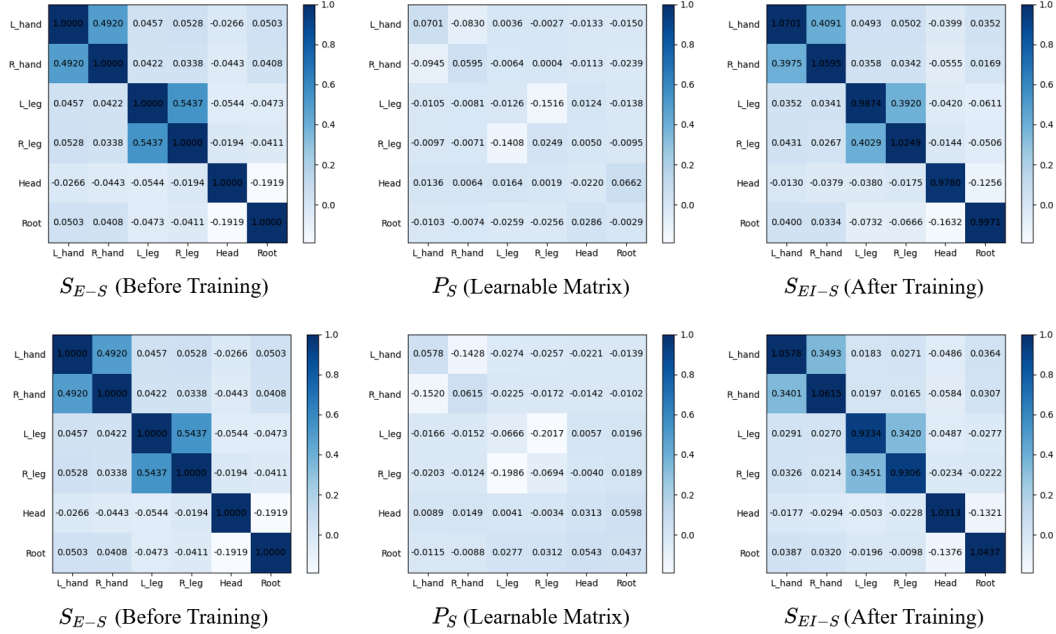

Figure 6: The visualization of $S_{E-S}$, $P_S$ and $S_{EI-S}$. First column: $S_{E-S}$ obtained using the AMASS dataset before training. First row: results from our model trained on the AMASS, DIP-IMU dataset. Second row: results from our model trained on the AnDy, CIP, and Emokine datasets.

- The learnable matrix $P_S$ adds small offsets to the spatial structure matrix, i.e., slightly suppresses the correlation between two hands, two legs and head vs. root. We attribute this to the different motion distributions among datasets: i) in AMASS, daily actions (e.g., walking, jogging, running, sitting and stretching) are dominant, and the movements of both hands and legs show extremely high consistency; ii) in the DIP-IMU, although daily actions are also the majority, there are a large number of single-hand and single-leg movements, such as single-hand raising, grasping and swinging; single-leg lifting, etc., which weaken the movement consistency of both hands and legs; iii) in the Andy and CIP, there are numerous industry-oriented activities, which are very different from daily movements, resulting in a relatively large adjustment range of the learnable matrix $P_S$. This demonstrates the effectiveness of our $P_S$ in adapting the structure matrix to different datasets, and maintains a high generalization ability.

## 4.5 Live Demo

We have implemented a real-time pose estimation visualization system using Python and Unity. We select a variety of actions to evaluate the performance of our model in real-world scenarios. These include everyday actions like walking, sitting, kicking, stretching, sports and more. Additionally, we choose some challenging movements such as single-leg standing, rolling, Chinese Kung Fu and dance movements to assess the generalization ability of our model. Through the live demo, it can be observed that even under intense physical activity, our approach still maintains long-term stability and shows robust generalization capability. Please refer to the **supplementary video** for our demo.

## 5 Conclusion

In this paper, we propose a novel sequence structure learning and modulation approach that empowers Transformers with the ability to model and utilize fixed-length sequence structural information. We present a simple yet effective Sequence Structure Module (SSM) to accomplish this, achieving impressive performance in sparse inertial pose estimation tasks. This showcases the powerful potential of the SSM to generalize to other Transformer-based fixed-length sequence tasks.

## 6  Acknowledgments

This work is supported by National Natural Science Foundation of China (62072383, 61702433), the Public Technology Service Platform Project of Xiamen City (No.3502Z20231043), Xiaomi Young Talents Program/Xiaomi Foundation, the Fundamental Research Funds for the Central Universities. This work is also partially supported by Royal Society (IEC\NSFC \211022).

## Footnotes

*Corresponding author (guoshihui@xmu.edu.cn) †Also with Key Laboratory of Digital Protection and Intelligent Processing of Intangible Cultural Heritage of Fujian and Taiwan, Ministry of Culture and Tourism.

## References

[1] Jimmy Lei Ba, Jamie Ryan Kiros, and Geoffrey E Hinton. Layer normalization. *arXiv preprint arXiv:1607.06450*, 2016.

[2] Shaojie Bai, J Zico Kolter, and Vladlen Koltun. An empirical evaluation of generic convolutional and recurrent networks for sequence modeling. *arXiv preprint arXiv:1803.01271*, 2018.

[3] Yiming Bao, Xu Zhao, and Dahong Qian. Fusepose: Imu-vision sensor fusion in kinematic space for parametric human pose estimation. *IEEE Transactions on Multimedia*, 2022.

[4] Iz Beltagy, Matthew E Peters, and Arman Cohan. Longformer: The long-document transformer. *arXiv preprint arXiv:2004.05150*, 2020.

[5] Angela Castillo, Maria Escobar, Guillaume Jeanneret, Albert Pumarola, Pablo Arbeláez, Ali Thabet, and Artsiom Sanakoyeu. Bodiffusion: Diffusing sparse observations for full-body human motion synthesis. In *Proceedings of the IEEE/CVF International Conference on Computer Vision*, pages 4221–4231, 2023.

[6] Anjun Chen, Xiangyu Wang, Shaohao Zhu, Yanxu Li, Jiming Chen, and Qi Ye. mmbody benchmark: 3d body reconstruction dataset and analysis for millimeter wave radar. In *Proceedings of the 30th ACM International Conference on Multimedia*, pages 3501–3510, 2022.

[7] Xiaowei Chen, Xiao Jiang, Jiawei Fang, Shihui Guo, Juncong Lin, Minghong Liao, Guoliang Luo, and Hongbo Fu. Dispad: Flexible on-body displacement of fabric sensors for robust joint-motion tracking. *Proceedings of the ACM on Interactive, Mobile, Wearable and Ubiquitous Technologies*, 7(1):1–27, 2023.

[8] Yilun Chen, Zhicheng Wang, Yuxiang Peng, Zhiqiang Zhang, Gang Yu, and Jian Sun. Cascaded pyramid network for multi-person pose estimation. In *Proceedings of the IEEE conference on computer vision and pattern recognition*, pages 7103–7112, 2018.

[9] Julia F. Christensen, Andrés Fernández, Rebecca A. Smith, Georgios Michalareas, Sina H. N. Yazdi, Fahima Farahi, Eva-Madeleine Schmidt, Nasimeh Bahmanian, and Gemma Roig. Emokinedataset, June 2023.

[10] Zihang Dai, Zhilin Yang, Yiming Yang, Jaime Carbonell, Quoc V Le, and Ruslan Salakhutdinov. Transformer-xl: Attentive language models beyond a fixed-length context. *arXiv preprint arXiv:1901.02860*, 2019.

[11] Jacob Devlin, Ming-Wei Chang, Kenton Lee, and Kristina Toutanova. Bert: Pre-training of deep bidirectional transformers for language understanding. *arXiv preprint arXiv:1810.04805*, 2018.

[12] Alexey Dosovitskiy, Lucas Beyer, Alexander Kolesnikov, Dirk Weissenborn, Xiaohua Zhai, Thomas Unterthiner, Mostafa Dehghani, Matthias Minderer, Georg Heigold, Sylvain Gelly, et al. An image is worth 16x16 words: Transformers for image recognition at scale. *arXiv preprint arXiv:2010.11929*, 2020.

[13] Yuming Du, Robin Kips, Albert Pumarola, Sebastian Starke, Ali Thabet, and Artsiom Sanakoyeu. Avatars grow legs: Generating smooth human motion from sparse tracking inputs with diffusion model. In *Proceedings of the IEEE/CVF Conference on Computer Vision and Pattern Recognition*, pages 481–490, 2023.

[14] Patric Eichelberger, Matteo Ferraro, Ursina Minder, Trevor Denton, Angela Blasimann, Fabian Krause, and Heiner Baur. Analysis of accuracy in optical motion capture–a protocol for laboratory setup evaluation. *Journal of biomechanics*, 49(10):2085–2088, 2016.

[15] Tamar Flash and Neville Hogan. The coordination of arm movements: an experimentally confirmed mathematical model. *Journal of neuroscience*, 5(7):1688–1703, 1985.

[16] Gutemberg Guerra-Filho. Optical motion capture: Theory and implementation. *RITA*, 12(2):61–90, 2005.

[17] Kaiming He, Xiangyu Zhang, Shaoqing Ren, and Jian Sun. Deep residual learning for image recognition. In *Proceedings of the IEEE conference on computer vision and pattern recognition*, pages 770–778, 2016.

[18] Yinghao Huang, Manuel Kaufmann, Emre Aksan, Michael J Black, Otmar Hilliges, and Gerard Pons-Moll. Deep inertial poser: Learning to reconstruct human pose from sparse inertial measurements in real time. *ACM Transactions on Graphics (TOG)*, 37(6):1–15, 2018.

[19] Jiaxi Jiang, Paul Streli, Manuel Meier, Andreas Fender, and Christian Holz. Egoposer: Robust real-time ego-body pose estimation in large scenes. *arXiv preprint arXiv:2308.06493*, 2023.

[20] Yifeng Jiang, Yuting Ye, Deepak Gopinath, Jungdam Won, Alexander W Winkler, and C Karen Liu. Transformer inertial poser: Real-time human motion reconstruction from sparse imus with simultaneous terrain generation. In *SIGGRAPH Asia 2022 Conference Papers*, pages 1–9, 2022.

[21] Nikita Kitaev, Łukasz Kaiser, and Anselm Levskaya. Reformer: The efficient transformer. *arXiv preprint arXiv:2001.04451*, 2020.

[22] Muhammed Kocabas, Chun-Hao P Huang, Otmar Hilliges, and Michael J Black. Pare: Part attention regressor for 3d human body estimation. In *Proceedings of the IEEE/CVF International Conference on Computer Vision*, pages 11127–11137, 2021.

[23] Kazutaka Kurihara, Shin'ichiro Hoshino, Katsu Yamane, and Yoshihiko Nakamura. Optical motion capture system with pan-tilt camera tracking and real time data processing. In *Proceedings 2002 IEEE international conference on robotics and automation (Cat. No. 02CH37292)*, volume 2, pages 1241–1248. IEEE, 2002.

[24] Shiyang Li, M Ziyang, Yingbo Zhou, Denny Wu, Qianxiao Li, Benjamin A Rozario, and Alexander Schwing. Enhancing the locality and breaking the memory bottleneck of transformer on time series forecasting. *arXiv preprint arXiv:1907.00235*, 2019.

[25] Wenhao Li, Mengyuan Liu, Hong Liu, Pichao Wang, Jialun Cai, and Nicu Sebe. Hourglass tokenizer for efficient transformer-based 3d human pose estimation. *arXiv preprint arXiv:2311.12028*, 2023.

[26] Zhen Liang, Dongquan Zhang, Guanghua Xu, Fangting Xie, Hao Guo, Jingyuan Cheng, et al. Smart garment: A long-term feasible, whole-body textile pressure sensing system. *IEEE Sensors Journal*, 2023.

[27] Bryan Lim, Sercan O Arik, Nicolas Loeff, and Tomas Pfister. Temporal fusion transformers for interpretable multi-horizon time series forecasting. *arXiv preprint arXiv:1912.09363*, 2021.

[28] Qiuxia Lin, Kerui Gu, Linlin Yang, and Angela Yao. Synthetic-to-real pose estimation with geometric reconstruction. *Advances in Neural Information Processing Systems*, 36, 2024.

[29] Matthew Loper, Naureen Mahmood, Javier Romero, Gerard Pons-Moll, and Michael J. Black. SMPL: A skinned multi-person linear model. *ACM Trans. Graphics (Proc. SIGGRAPH Asia)*, 34(6):248:1–248:16, October 2015.

[30] Ilya Loshchilov and Frank Hutter. Decoupled weight decay regularization. *arXiv preprint arXiv:1711.05101*, 2017.

[31] Jianxin Ma, Peng Kang, Xiaojie Wang, Mengshuo Wang, Weinan Zhang, and Yong Yu. Logtrans: A log sparse transformer network for sequential recommendation. *Proceedings of the 43rd International ACM SIGIR Conference on Research and Development in Information Retrieval*, pages 2347–2356, 2020.

[32] Naureen Mahmood, Nima Ghorbani, Nikolaus F Troje, Gerard Pons-Moll, and Michael J Black. Amass: Archive of motion capture as surface shapes. In *Proceedings of the IEEE/CVF international conference on computer vision*, pages 5442–5451, 2019.

[33] Pauline Maurice, Adrien Malaisé, Clélie Amiot, Nicolas Paris, Guy-Junior Richard, Olivier Rochel, and Serena Ivaldi. Human movement and ergonomics: An industry-oriented dataset for collaborative robotics. *The International Journal of Robotics Research*, 38(14):1529–1537, 2019.

[34] Vimal Mollyn, Riku Arakawa, Mayank Goel, Chris Harrison, and Karan Ahuja. Imuposer: Full-body pose estimation using imus in phones, watches, and earbuds. In *Proceedings of the 2023 CHI Conference on Human Factors in Computing Systems*, pages 1–12, 2023.

[35] Md Moniruzzaman, Zhaozheng Yin, Md Sanzid Bin Hossain, Hwan Choi, and Zhishan Guo. Wearable motion capture: Reconstructing and predicting 3d human poses from wearable sensors. *IEEE Journal of Biomedical and Health Informatics*, 2023.

[36] L Noitom. Perception neuron. 2017.

[37] Manuel Palermo, Sara M Cerqueira, João André, António Pereira, and Cristina P Santos. From raw measurements to human pose-a dataset with low-cost and high-end inertial-magnetic sensor data. *Scientific Data*, 9(1):591, 2022.

[38] Shaohua Pan, Qi Ma, Xinyu Yi, Weifeng Hu, Xiong Wang, Xingkang Zhou, Jijunnan Li, and Feng Xu. Fusing monocular images and sparse imu signals for real-time human motion capture. *arXiv preprint arXiv:2309.00310*, 2023.

[39] Hui En Pang, Zhongang Cai, Lei Yang, Qingyi Tao, Zhonghua Wu, Tianwei Zhang, and Ziwei Liu. Towards robust and expressive whole-body human pose and shape estimation. *Advances in Neural Information Processing Systems*, 36, 2024.

[40] Adam Paszke, Sam Gross, Francisco Massa, Adam Lerer, James Bradbury, Gregory Chanan, Trevor Killeen, Zeming Lin, Natalia Gimelshein, Luca Antiga, et al. Pytorch: An imperative style, high-performance deep learning library. *Advances in neural information processing systems*, 32, 2019.

[41] Monique Paulich, Martin Schepers, Nina Rudigkeit, and Giovanni Bellusci. Xsens mtw awinda: Miniature wireless inertial-magnetic motion tracker for highly accurate 3d kinematic applications. *Xsens: Enschede, The Netherlands*, pages 1–9, 2018.

[42] Natural Point. Optitrack. *Natural Point, Inc*, 2011.

[43] Jose Luis Ponton, Haoran Yun, Andreas Aristidou, Carlos Andujar, and Nuria Pelechano. Sparseposer: Real-time full-body motion reconstruction from sparse data. *ACM Transactions on Graphics*, 2023.

[44] Ke Sun, Bin Xiao, Dong Liu, and Jingdong Wang. Deep high-resolution representation learning for human pose estimation. In *Proceedings of the IEEE/CVF conference on computer vision and pattern recognition*, pages 5693–5703, 2019.

[45] Zhenhua Tang, Zhaofan Qiu, Yanbin Hao, Richang Hong, and Ting Yao. 3d human pose estimation with spatio-temporal criss-cross attention. In *Proceedings of the IEEE/CVF Conference on Computer Vision and Pattern Recognition*, pages 4790–4799, 2023.

[46] Matt Trumble, Andrew Gilbert, Charles Malleson, Adrian Hilton, and John Collomosse. Total capture: 3d human pose estimation fusing video and inertial sensors. In *2017 British Machine Vision Conference (BMVC)*, 2017.

[47] Ashish Vaswani, Noam Shazeer, Niki Parmar, Jakob Uszkoreit, Llion Jones, Aidan N Gomez, Ł ukasz Kaiser, and Illia Polosukhin. Attention is all you need. In I. Guyon, U. Von Luxburg, S. Bengio, H. Wallach, R. Fergus, S. Vishwanathan, and R. Garnett, editors, *Advances in Neural Information Processing Systems*, volume 30. Curran Associates, Inc., 2017.

[48] Timo Von Marcard, Bodo Rosenhahn, Michael J Black, and Gerard Pons-Moll. Sparse inertial poser: Automatic 3d human pose estimation from sparse imus. In *Computer graphics forum*, volume 36, pages 349–360. Wiley Online Library, 2017.

[49] Sinong Wang, Belinda Z Li, Madian Khabsa, Han Fang, and Hao Ma. Linformer: Self-attention with linear complexity. *arXiv preprint arXiv:2006.04768*, 2020.

[50] Haixu Wu, Jiehui Xu, Jianmin Wang, and Mingsheng Long. Autoformer: Decomposition transformers with auto-correlation for long-term series forecasting. *Advances in Neural Information Processing Systems*, 34:22419–22430, 2021.

[51] Yufei Xu, Jing Zhang, Qiming Zhang, and Dacheng Tao. Vitpose: Simple vision transformer baselines for human pose estimation. *Advances in Neural Information Processing Systems*, 35:38571–38584, 2022.

[52] Haoyong Xue, Duanfei Zhang, Jie Cao, Chang Liu, Xincheng Yao, Xiyang Zhang, Yikang Shen, Yue Zhang, and Xinyu Dai. Pyraformer: Low-complexity pyramidal attention for long-range time series modeling and forecasting. *arXiv preprint arXiv:2106.00204*, 2021.

[53] Hongfei Xue, Qiming Cao, Yan Ju, Haochen Hu, Haoyu Wang, Aidong Zhang, and Lu Su. M4esh: mmwave-based 3d human mesh construction for multiple subjects. In *Proceedings of the 20th ACM Conference on Embedded Networked Sensor Systems*, pages 391–406, 2022.

[54] Sen Yang, Zhibin Quan, Mu Nie, and Wankou Yang. Transpose: Keypoint localization via transformer. In *Proceedings of the IEEE/CVF International Conference on Computer Vision*, pages 11802–11812, 2021.

[55] Xinyu Yi, Yuxiao Zhou, Marc Habermann, Soshi Shimada, Vladislav Golyanik, Christian Theobalt, and Feng Xu. Physical inertial poser (pip): Physics-aware real-time human motion tracking from sparse inertial sensors. In *Proceedings of the IEEE/CVF Conference on Computer Vision and Pattern Recognition*, pages 13167–13178, 2022.

[56] Xinyu Yi, Yuxiao Zhou, and Feng Xu. Transpose: Real-time 3d human translation and pose estimation with six inertial sensors. *ACM Transactions on Graphics (TOG)*, 40(4):1–13, 2021.

[57] Xinyu Yi, Yuxiao Zhou, and Feng Xu. Physical non-inertial poser (pnp): Modeling non-inertial effects in sparse-inertial human motion capture. In *SIGGRAPH 2024 Conference Papers*, 2024.

[58] Bruce XB Yu, Zhi Zhang, Yongxu Liu, Sheng-hua Zhong, Yan Liu, and Chang Wen Chen. Gla-gcn: Global-local adaptive graph convolutional network for 3d human pose estimation from monocular video. In *Proceedings of the IEEE/CVF International Conference on Computer Vision*, pages 8818–8829, 2023.

[59] George Zerveas, Sercan O Arik, Rajiv R Raitoharju, and Tomas Pfister. A transformer-based framework for multivariate time series representation learning. *arXiv preprint arXiv:2106.01770*, 2021.

[60] Yu Zhang, Songpengcheng Xia, Lei Chu, Jiarui Yang, Qi Wu, and Ling Pei. Dynamic inertial poser (dynaip): Part-based motion dynamics learning for enhanced human pose estimation with sparse inertial sensors. *IEEE / CVF Computer Vision and Pattern Recognition Conference (CVPR)*, 2024.

[61] Qitao Zhao, Ce Zheng, Mengyuan Liu, and Chen Chen. A single 2d pose with context is worth hundreds for 3d human pose estimation. *arXiv preprint arXiv:2311.03312*, 2023.

[62] Jianan Zhen, Qi Fang, Jiaming Sun, Wentao Liu, Wei Jiang, Hujun Bao, and Xiaowei Zhou. Smap: Single-shot multi-person absolute 3d pose estimation. In *Computer Vision–ECCV 2020: 16th European Conference, Glasgow, UK, August 23–28, 2020, Proceedings, Part XV 16*, pages 550–566. Springer, 2020.

[63] Xiaozheng Zheng, Zhuo Su, Chao Wen, Zhou Xue, and Xiaojie Jin. Realistic full-body tracking from sparse observations via joint-level modeling. In *Proceedings of the IEEE/CVF International Conference on Computer Vision*, pages 14678–14688, 2023.

[64] Haoyi Zhou, Shanghang Zhang, Jieqi Peng, Shuai Zhang, Jianxin Li, Hui Xiong, and Wancai Zhang. Informer: Beyond efficient transformer for long sequence time-series forecasting. *Proceedings of the AAAI Conference on Artificial Intelligence*, 35(12):11106–11115, 2021.

[65] Tian Zhou, Ziqing Ma, Qingsong Wen, Xue Wang, Liang Sun, and Rong Jin. Fedformer: Frequency enhanced decomposed transformer for long-term series forecasting. *arXiv preprint arXiv:2201.12740*, 2022.

[66] Yi Zhou, Connelly Barnes, Jingwan Lu, Jimei Yang, and Hao Li. On the continuity of rotation representations in neural networks. In *Proceedings of the IEEE/CVF conference on computer vision and pattern recognition*, pages 5745–5753, 2019.

[67] Wentao Zhu, Xiaoxuan Ma, Zhaoyang Liu, Libin Liu, Wayne Wu, and Yizhou Wang. Motion-bert: A unified perspective on learning human motion representations. In *Proceedings of the IEEE/CVF International Conference on Computer Vision*, pages 15085–15099, 2023.

[68] Chengxu Zuo, Jiawei Fang, Shihui Guo, and Yipeng Qin. Self-adaptive motion tracking against on-body displacement of flexible sensors. In *Thirty-seventh Conference on Neural Information Processing Systems*, 2023.

[69] Chengxu Zuo, Yiming Wang, Lishuang Zhan, Shihui Guo, Xinyu Yi, Feng Xu, and Yipeng Qin. Loose inertial poser: Motion capture with imu-attached loose-wear jacket. *IEEE / CVF Computer Vision and Pattern Recognition Conference (CVPR)*, 2024.

# Appendix

## 7 Detailed Network Architecture

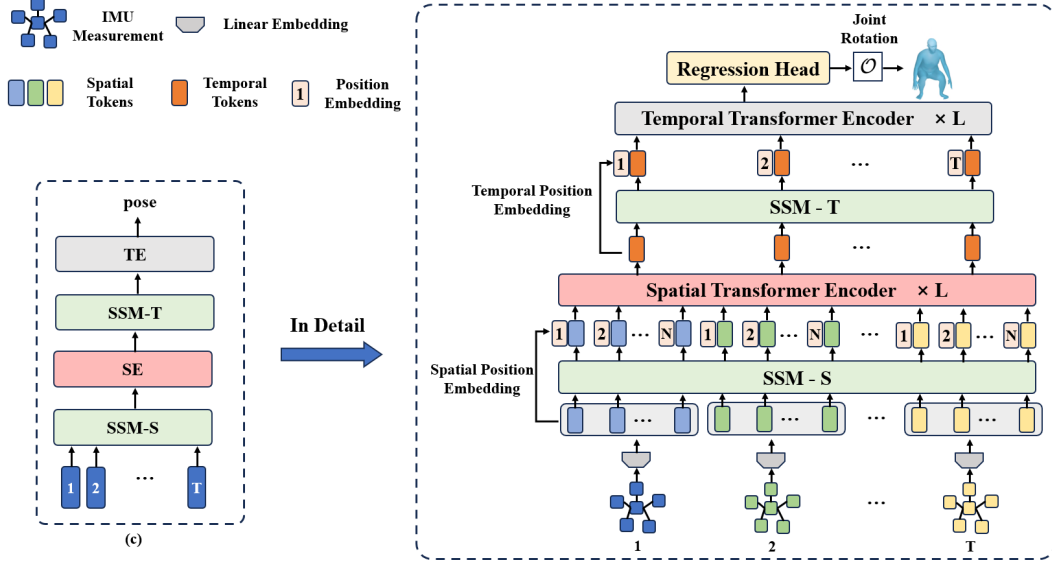

Figure 7: Our network architecture in detail.

As shown in Fig. 7, our network mainly consists of Linear Embedding, SSM-S, Spatial Transformer Encoder, SSM-T, Temporal Transformer Encoder and Regression Head Module. In the main text, we introduce SSM-S and SSM-T. Here, we provide an introduction to the remaining parts.

**Linear Embedding**. Our network takes $\mathcal{M} \in \mathbb{R}^{T \times N \times 12}$, the measurements from $N$ IMUs over the past $T$ frames as input. Firstly, we linearly map each IMU measurement $m_{(i,j)} \in \mathbb{R}^{12}$ into an embedding vector $z_{(i,j)} \in \mathbb{R}^D$ by means of a learnable matrix $E \in \mathbb{R}^{12 \times D}$:

$$z_{(i,j)} = m_{(i,j)}E \tag{9}$$

After that, the input $\mathcal{M} \in \mathbb{R}^{T \times N \times 12}$ becomes $F_s \in \mathbb{R}^{T \times N \times D}$. The spatial feature $F_s$ is fed into SSM-S to inject spatial structure, and the resulting $\widetilde{F_s}$ is then send into the Spatial Transformer Encoder to model the motion correlation between body joints.

**Spatial Transformer Encoder**. Given $\widetilde{F_s}$, we first add a learnable spatial position embedding $P_s \in \mathbb{R}^{N \times D}$ to each token for maintaining spatial position information. The resulting joint sequence of features $z_s$ are fed into a spatial encoder consisting of a sequence of $L$ transformer layers. Each layer $\ell$ comprises of Multi-Head Self-Attention [47], LayerNorm [1], and MLP blocks:

$$z_s = \widetilde{F_s} + P_s \tag{10}$$
$$y_s^\ell = \text{MSA}(\text{LN}(z_s^\ell)) + z_s^\ell \tag{11}$$
$$z_s^{\ell+1} = \text{MLP}(\text{LN}(y_s^\ell)) + y_s^\ell \tag{12}$$

The output of the last transformer layer is $z_s^L \in \mathbb{R}^{T \times N \times D}$, which is sent into SSM-T where temporal structural information is incorporated.

**Temporal Transformer Encoder**. We treat the output of SSM-T, $\widetilde{F_t} \in \mathbb{R}^{T \times N \times D}$, as the input of temporal transformer encoder, to further extract the global dependencies across frames in the entire sequence. We first reshape $\widetilde{F_t}$ into $\widetilde{F_t} \in \mathbb{R}^{T \times (N \cdot D)}$. Before the temporal transformer encoder, we add a learnable temporal positional embedding $P_t \in \mathbb{R}^{T \times (N \cdot D)}$ to retain frame position information. The resulting frame sequence of features $z_t$ are fed into a temporal encoder, which has the same architecture as the spatial transformer encoder. The procedure can be formulated as:

$$z_t = \widetilde{F_t} + P_t \tag{13}$$

$$y_t^\ell = \text{MSA}(\text{LN}(z_t^\ell)) + z_t^\ell \tag{14}$$

$$z_t^{\ell+1} = \text{MLP}(\text{LN}(y_t^\ell)) + y_t^\ell \tag{15}$$

The output of the last transformer layer is $z_t^L \in \mathbb{R}^{T \times (N \cdot D)}$, a compact spatial-temporal feature representation, which is sent into regression head module.

**Regression Head Module.** We map $z_t^L$ into the whole body joint rotations $\hat{\mathcal{O}} \in \mathbb{R}^{T \times J \times 6}$ using Layer Normalization [1] and MLP block:

$$\hat{\mathcal{O}} = \text{MLP}(\text{LN}(z_t^L)) \tag{16}$$

where $J$ denotes the number of joints and 6 denotes 6D rotation representation [66]. The whole network is optimized by minimizing the Mean Squared Error (MSE) between $\hat{\mathcal{O}}$ and the ground-truth $\mathcal{O}$ as:

$$\mathcal{L} = \left\| \hat{\mathcal{O}} - \mathcal{O} \right\|^2 \tag{17}$$

During the training stage, we compute the loss using the predictions $\hat{\mathcal{O}}$ and ground-truth $\mathcal{O}$ of $T$ frames. During the inference stage, we only utilize the last frame as the output.

# 8 Additional Experiments

**Result of Another Definition for $S_{E-T}$.**

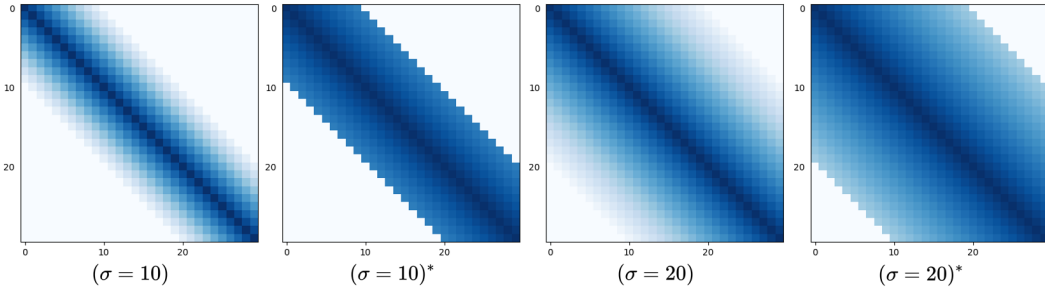

$$(\sigma = 10) \qquad (\sigma = 10)^* \qquad (\sigma = 20) \qquad (\sigma = 20)^*$$

Figure 8: The visualization of $S_{E-T}$ of different definition when $\sigma = 10$ and $\sigma = 20$.

We further explore an alternative definition of $S_{E-T}$:

$$S_{E-T}(i,j) = \begin{cases} 0 & \text{if } |i-j| \geq \sigma, \\ 1 - \dfrac{|i-j|}{T} & \text{else} \end{cases} \tag{18}$$

Differing from Eq. (8), in Eq. (18), $\sigma$ has been replaced by $T$. We compare the cases with $\sigma = 10$ and $\sigma = 20$ to the previous scenario, denoted as $*$. The visualization for $S_{E-T}$ are shown in Fig. 8 and the performance on TotalCapture dataset is shown in Tab. 6. It is easily discernible that the definition of Eq. (8) is significantly superior to that of Eq. (18), both in terms of accuracy and stability.

Table 6: The performance under different definition for $S_{E-T}$ on TotalCapture dataset.

|  | Ang Err | Jitter | $\tau$ |
|---|---|---|---|
| $(\sigma = 10)$ | **6.82** | **0.09** | **7.46** |
| $(\sigma = 10)^*$ | 7.86 | 0.08 | 8.51 |
| $(\sigma = 20)$ | 7.76 | 0.07 | 8.32 |
| $(\sigma = 20)^*$ | 8.33 | 0.07 | 8.93 |

**Ablation Study of the Order of SE/TE.**

We provide the results of switching the order from "SE-TE" to "TE-SE" as Tab. 7. The results show that our "SE-TE" order outperforms its "TE-SE" variant by a large margin, which is consistent with the common practice in many spatio-temporal frameworks.

Table 7: The performance under different definition for $S_{E-T}$ on TotalCapture dataset.

| Method | Ang Err | Jitter | $\tau$ |
|---|---|---|---|
| SE-TE (ours) | **6.82** | **0.09** | **7.46** |
| TE-SE | 8.67 | 0.11 | 9.68 |

**Generalization of Module under More Sensors**.

Although we focused on the more challenging "sparse" settings (6 IMUs) in our paper, showing additional results of applying our modules to more sensors could further demonstrate their generalization ability. Specifically, we increase the number of IMUs from 6 to 10 (with the additional 4 IMUs placed on the left and right shoulders and thighs), and conduct experiments and ablation studies on the DIP-IMU dataset.

Table 8: The performance under more sensors on DIP-IMU dataset.

| IMUs | Ang Err | Jitter | $\tau$ |
|---|---|---|---|
| 6 (ours) | 6.06 | 0.07 | 6.49 |
| 10 (ours) | **4.40** | **0.03** | **4.53** |
| 10 (ours w/o SSM-S) | 5.69 | 0.03 | 5.86 |
| 10 (ours w/o SSM-T) | 4.56 | 0.13 | 5.19 |
| 10 (ours w/o SSM-S, w/o SSM-T) | 5.58 | 0.14 | 6.42 |

As shown in Tab. 8, after adding 4 IMUs, our proposed method still works effectively (higher accuracy and lower jitter). Additionally, the ablation study results show that the roles of SSM-S and SSM-T remain consistent with their performance when using 6 IMUs. That is, SSM-S improves the accuracy of motion prediction, while SSM-T reduces jitter to enhance the coherence of the posture. Based on the above experimental results, it can be concluded that our proposed method maintains strong generalization ability as the number of sensors increases.

**Performance on Different Users with Different Physiques**. To study the performance on different users with different physiques, we conduct experiments on DIP-IMU dataset. To demonstrate this more clearly, we computed the BMI values (BMI = Mass(kg) / Height$^2$(m)) for each individual, categorize them into three groups, and report our model's performance across these three categories as Tab. 10:

Table 9: The performance on different users with different physiques.

| BMI | ID | Number | Ang Err | Jitter | $\tau$ |
|---|---|---|---|---|---|
| C1: BMI<21.7 | S2/S6 | 2 | 7.33 | 0.08 | 7.94 |
| C2: 21.7<=BMI<=24.9 | S1/S3/S5/S7/S8/S9 | 6 | 7.63 | 0.09 | 8.35 |
| C3: BMI>24.9 | S4/S10 | 2 | 8.07 | 0.09 | 8.83 |
| average | / | 10 | 7.66 | 0.09 | 8.35 |

The experimental results demonstrate that our method is robust and performs well across users with different physiques. For reference, we also include the results for each individual as Tab. 10.

**More Comparisons with Other NN Structures**. As shown in Tab. 11, we construct a spatio-temporal framework using GCN layers as the spatial encoder and Conv1d layers as the temporal encoder. The results show that our method significantly outperforms the GCN + Conv1d implementation.

# 9   Discussion

**The Design of** $\tau$. In our experiments, we observed that it is difficult for *Ang Err* and *Jitter* to simultaneously reach their minima, forming a trade-off in-between. For example, in Tab. 6, with

Table 10: The performance for each individual on different users with different physiques.

| ID | Ang Err | Jitter | $\tau$ | Mass(kg) | Height(cm) | BMI | Categorya |
|----|---------|--------|--------|----------|------------|-----|-----------|
| s1 | 7.74 | 0.05 | 8.13 | 86 | 186 | 24.85 | 2 |
| s2 | 7.47 | 0.05 | 7.85 | 65 | 178 | 20.51 | 1 |
| s3 | 7.95 | 0.11 | 8.87 | 87 | 187 | 24.87 | 2 |
| s4 | 7.63 | 0.11 | 8.51 | 78 | 170 | 26.98 | 3 |
| s5 | 7.64 | 0.12 | 8.61 | 80 | 180 | 24.69 | 2 |
| s6 | 7.20 | 0.11 | 8.03 | 58 | 172 | 19.60 | 1 |
| s7 | 6.89 | 0.09 | 7.53 | 70 | 178 | 22.09 | 2 |
| s8 | 7.13 | 0.09 | 7.80 | 80 | 180 | 24.69 | 2 |
| s9 | 8.48 | 0.06 | 9.00 | 85 | 187 | 24.30 | 2 |
| s10 | 8.51 | 0.07 | 9.12 | 87 | 181 | 26.55 | 3 |

Table 11: More Comparisons with Other NN Structures.

| Method | Ang Err | Jitter | $\tau$ |
|--------|---------|--------|--------|
| GCN + Conv1d | 14.31 | 0.28 | 18.93 |
| Transformer + Transformer(ours) | **6.82** | **0.09** | **7.46** |

$\sigma = 20$, *Ang Err* is 7.76 and *Jitter* is 0.07; with $\sigma = 10$, *Ang Err* decreases to 6.82 while *Jitter* increases to 0.09. To strike a balance in this trade-off, we introduce $\tau = (AngErr) * \text{Exp}(Jitter)$ to combine *Ang Err* and *Jitter* into a single measure. The reason for using $\text{Exp}$ is that we have observed that the impact of the *Jitter* on viewing experience is non-linear:

- When *Jitter* is relatively high (e.g., $jitter > 0.3$), the visual quality is unacceptable.
- When *Jitter* is relatively low (e.g., $0 < jitter < 0.2$), the visual experience is good and insensitive to changes in *Jitter*.

Therefore, we use $\text{Exp}$ to measure the impact of *Jitter* on the viewing experience.

## 10 Limitation

**Lack of Global Translation**. Although our method has made significant progress in predicting human pose from sparse inertial sensor data, it lacks global tracking of human motion trajectories. We believe that relying solely on IMU for accurate global translation prediction is challenging, as IMUs may drift, resulting in unreliable measurements of acceleration. Combining other types of sensors with IMUs is a promising solution.

**The Construction for Explicit Structure**. As mentioned earlier, any matrix can serve as an explicit structure $S_E$. We provide two examples demonstrating how to construct an explicit structure solely for illustrative purposes. Constructing $S_E$ more effectively for specific tasks requires further exploration.

